# A NOVEL NET THAT LEARNS
# SEQUENTIAL DECISION PROCESS

G.Z. SUN, Y.C. LEE and H.H. CHEN

*Department of Physics and Astronomy*
*and*
*Institute for Advanced Computer Studies*
**UNIVERSITY OF MARYLAND,COLLEGE PARK,MD 20742**

## ABSTRACT

We propose a new scheme to construct neural networks to classify patterns. The new scheme has several novel features :

1. We focus attention on the important attributes of patterns in ranking order. Extract the most important ones first and the less important ones later.

2. In training we use the information as a measure instead of the error function.

3. A multi-perceptron-like architecture is formed auomatically. Decision is made according to the tree structure of learned attributes.

This new scheme is expected to self-organize and perform well in large scale problems.

# 1 INTRODUCTION

It is well known that two-layered perceptron with binary connections but no hidden units is unsuitable as a classifier due to its limited power [1]. It cannot solve even the simple *exclusive-or* problem. Two extensions have been proposed to remedy this problem. The first is to use higher order connections [2]. It has been demonstrated that high order connections could in many cases solve the problem with speed and high accuracy [3], [4]. The representations in general are more local than distributive. The main drawback is however the combinatorial explosion of the number of high-order terms. Some kind of heuristic judgement has to be made in the choice of these terms to be represented in the network.

A second proposal is the multi-layered binary network with hidden units [5]. These hidden units function as features extracted from the bottom input layer to facilitate the classification of patterns by the output units. In order to train the weights, learning algorithms have been proposed that back-propagate the errors from the visible output layer to the hidden layers for eventual adaptation to the desired values. The multi-layered networks enjoy great popularity in their flexibility.

However, there are also problems in implementing the multi-layered nets. Firstly, there is the problem of allocating the resources. Namely, how many hidden units would be optimal for a particular problem. If we allocate too many, it is not only wasteful but also could negatively affect the performance of the network. Since too many hidden units implies too many free parameters to fit specifically the training patterns. Their ability to generalize to noval test patterns would be adversely affected. On the other hand, if too few hidden units were allocated then the network would not have the power even to represent the trainig set. How could one judge beforehand how many are needed in solving a problem? This is similar to the problem encountered in the high order net in its choice of high order terms to be represented.

Secondly, there is also the problem of scaling up the network. Since the network represents a parallel or coorperative process of the whole system, each added unit would interact with every other units. This would become a serious problem when the size of our patterns becomes large.

Thirdly, there is no sequential communication among the patterns in the conventional network. To accomplish a cognitive function we would need the patterns to interact and communicate with each other as the human reasoning does. It is difficult to envision such an interacton in current systems which are basically input-output mappings.

# 2 THE NEW SCHEME

In this paper, we would like to propose a scheme that constructs a network taking advantages of both the parallel and the sequential processes.

We note that in order to classify patterns, one has to extract the intrinsic features, which we call attributes. For a complex pattern set, there may be a large number of attributes. But differnt attributes may have different

ranking of importance. Instead of extracing them all simultaneously it may be wiser to extract them sequentially in order of its importance [6], [7]. Here the importance of an attribute is determined by its ability to partition the pattern set into sub-categories. A measure of this ability of a processing unit should be based on the extracted information. For simplicity, let us assume that there are only two categories so that the units have only binary output values 1 and 0 ( but the input patterns may have analog representations). We call these units, including their connection weights to the input layer, *nodes*. For given connection weights, the patterns that are classified by a *node* as in category 1 may have their true classifications either 1 or 0. Similarly, the patterns that are classified by a *node* as in category 0 may also have their true classifications either 1 or 0. As a result, four groups of patterns are formed: (1,1), (0,0), (1,0), (0,1). We then need to judge on the efficiency of the *node* by its ability to split these patterns optimally. To do this we shall construct the impurity fuctions for the *node*. Before splitting, the impurity of the input patterns reaching the node is given by

$$I_b = -P_1^b \log P_1^b - P_0^b \log P_0^b \tag{1}$$

where $P_1^b = N_1^b/N$ is the probability of being truely classified as in category 1, and $P_0^b = N_0^b/N$ is the probability of being truely classified as in category 0. After splitting, the patterns are channelled into two branches, the impurity becomes

$$I_a = -P_1^a \sum_{j=0,1} P(j,1) \log P(j,1) - P_0^a \sum_{j=0,1} P(j,0) \log P(j,0) \tag{2}$$

where $P_1^a = N_1^a/N$ is the probability of being classified by the node as in category 1, $P_0^a = N_0^b/N$ is the probability of being classified by the node as in category 0, and $P(j,i)$ is the probability of a pattern, which should be in category j, but is classified by the node as in category i. The difference

$$\Delta I = I_b - I_a \tag{3}$$

represents the decrease of the impurity at the node after splitting. It is the quantity that we seek to optimize at each node. The logarithm in the impurity function come from the information entropy of Shannon and Weaver. For all practical purpose, we found the optimization of (3) the same as maximizing the entropy [6]

$$S = \frac{N_1}{N}\left[(\frac{N_{01}}{N_1})^2 + (\frac{N_{11}}{N_1})^2\right] + \frac{N_0}{N}\left[(\frac{N_{00}}{N_0})^2 + (\frac{N_{10}}{N_0})^2\right] \tag{4}$$

where $N_i$ is the number of training patterns classified by the node as in category i, $N_{ij}$ is the number of training patterns with true classification in category i but classified by the node as in category j. Later we shall call the terms in the first bracket $S_1$ and the second $S_2$. Obviously, we have

$$N_i = N_{0i} + N_{1i}, \qquad i = 0,1$$

After we trained the first unit, the training patterns were split into two branches by the unit. If the classificaton in either one of these two branches is pure enough, or equivalently either one of $S_1$ and $S_2$ is fairly close to 1, then we would terminate that branch ( or branches ) as a leaf of the decision tree, and classify the patterns as such. On the other hand, if either branch is not pure enough, we add additional node to split the pattern set further. The subsequent unit is trained with only those patterns channeled through this branch. These operations are repeated until all the branches are terminated as leaves.

# 3   LEARNING ALGORITHM

We used the stochastic gradient descent method to learn the weights of each node. The training set for each node are those patterns being channeled to this node. As stated in the previous section, we seek to maximize the entropy function S. The learning of the weights is therefore conducted through

$$\Delta W_j = \eta \frac{\partial S}{\partial W_j} \tag{5}$$

Where $\eta$ is the learning rate. The gradient of S can be calculated from the following equation

$$\frac{\partial S}{\partial W_j} = \frac{1}{N}\Big[(1 - 2\frac{N_{01}^2}{N_1^2})\frac{\partial N_{11}}{\partial W_j} + (1 - 2\frac{N_{11}^2}{N_1^2})\frac{\partial N_{01}}{\partial W_j} +$$

$$(1 - 2\frac{N_{00}^2}{N_0^2})\frac{\partial N_{10}}{\partial W_j} + (1 - 2\frac{N_{10}^2}{N_0^2})\frac{\partial N_{00}}{\partial W_j}\Big] \tag{6}$$

Using analog units

$$O^r = \frac{1}{1 + exp(-\sum_j W_j I_j^r)} \tag{7}$$

we have

$$\frac{\partial O^r}{\partial W_j} = O^r(1 - O^r)I_j^r \tag{8}$$

Furthermore, let $A^r = 1$ or 0 being the true answer for the input pattern r , then

$$N_{ij} = \sum_{r=1}^{N}\Big[iA^r + (1 - i)(1 - A^r)\Big]\Big[jO^r + (1 - j)(1 - O^r)\Big] \tag{9}$$

Substituting these into equation (5), we get

$$\Delta W_j = 2\eta \sum_r \Big[2A^r(\frac{N_{11}}{N_1} - \frac{N_{10}}{N_0}) + \frac{N_{10}^2}{N_0^2} - \frac{N_{11}^2}{N_1^2}\Big]O^r(1 - O^r)I_j^r \tag{10}$$

In applying the formula (10),instead of calculating the whole summation at once, we update the weights for each pattern individually. Meanwhile we update $N_{ij}$ in accord with equation (9).

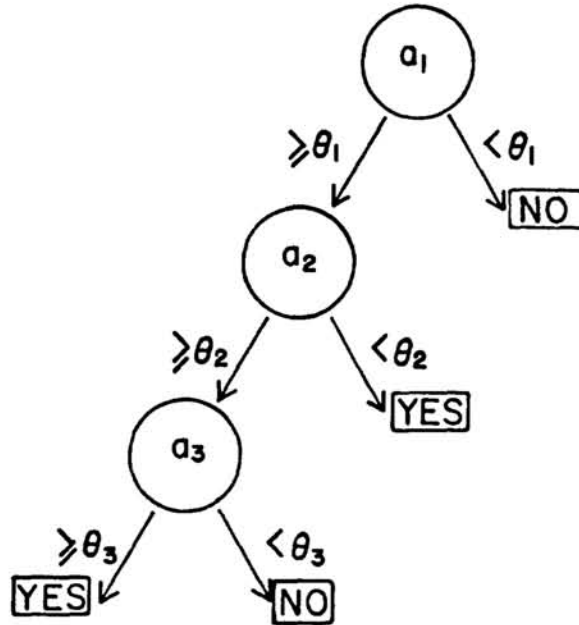

Figure 1: *The given classification tree, where $\theta_1, \theta_2$ and $\theta_3$ are chosen to be all zeros in the numerical example.*

# 4 AN EXAMPLE

To illustrate our method, we construct an example which is itself a decision tree. Assuming there are three hidden variables $a_1, a_2, a_3$, a pattern is given by a ten-dimensional vector $I_1, I_2, \ldots, I_{10}$, constructed from the three hidden variables as follows

$$
\begin{aligned}
I_1 &= a_1 + a_3 & I_6 &= 2a_3 \\
I_2 &= 2a_1 - a_2 & I_7 &= a_3 - a_1 \\
I_3 &= a_3 - 2a_2 & I_8 &= 2a_1 + 3a_3 \\
I_4 &= a_1 + 2a_2 + 3a_3 & I_9 &= 4a_3 - 3a_1 \\
I_5 &= 5a_1 - 4a_4 & I_{10} &= 2a_1 + 2a_2 + 2a_3.
\end{aligned}
$$

A given pattern is classified as either 1 (yes) or 0 (no) according to the corresponding values of the hidden variables $a_1, a_2, a_3$. The actual decision is derived from the decision tree in $Fig.1$.

In order to learn this classification tree, we construct a training set of 5000 patterns generated by randomly chosen values $a_1, a_2, a_3$ in the interval -1 to +1. We randomly choose the initial weights for each node, and terminate

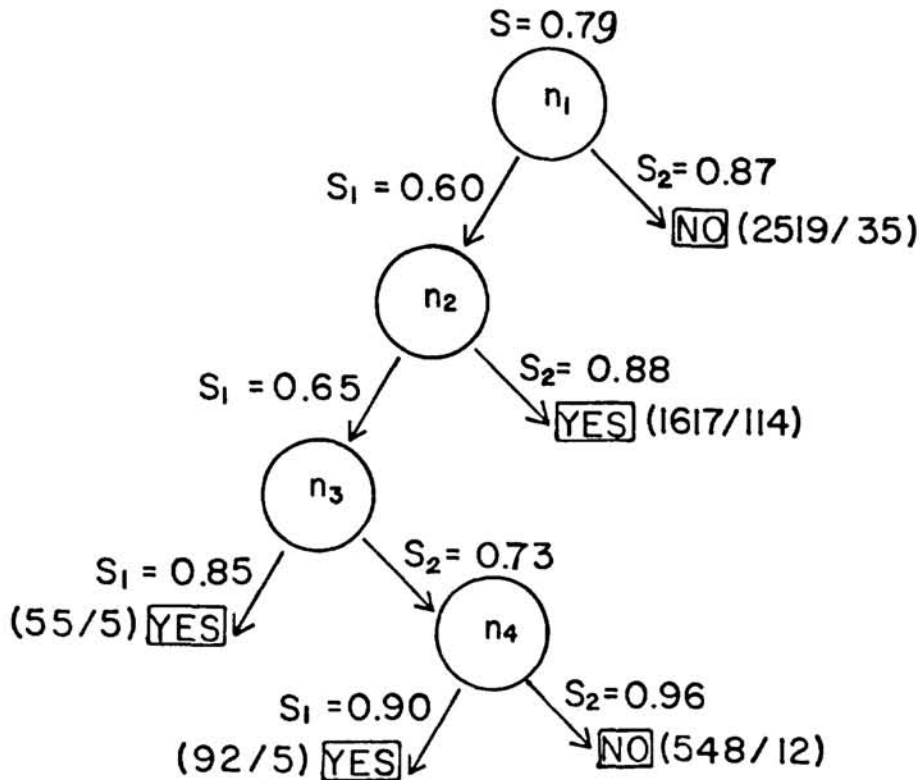

Figure 2: The learned classification tree structure

a branch as a leaf whenever the branch entropy is greater than 0.80. The entropy is started at $S = 0.65$, and terminated at its maximum value $S = 0.79$ for the first node. The two branches of this node have the entropy fuction valued at $S_1 = 0.61, S_2 = 0.87$ respectively. This corrosponds to 2446 patterns channeled to the first branch and 2554 to the second. Since $S_2 > 0.80$ we terminate the second branch. Among 2554 patterns channeled to the second branch there are 2519 patterns with true classification as *no* and 35 *yes* which are considered as errors. After completing the whole training process, there are totally four nodes automatically introduced. The final result is shown in a tree structure in *Fig*.2.

The total errors classified by the learned tree are 3.4 % of the 5000 trainig patterns. After trainig we have tested the result using 10000 novel patterns, the error among which is 3.2 %.

# 5 SUMMARY

We propose here a new scheme to construct neural network that can automatically learn the attributes sequentially to facilitate the classification of patterns according to the ranking importance of each attribute. This scheme uses information as a measure of the performance of each unit. It is

self-organized into a presumably *optimal* structure for a specific task. The sequential learning procedure focuses attention of the network to the most important attribute first and then branches out' to the less important attributes. This strategy of searching for attributes would alleviate the scale up problem forced by the overall parallel back-propagation scheme. It also avoids the problem of resource allocation encountered in the high-order net and the multi-layered net. In the example we showed the performance of the new method is satisfactory. We expect much better performance in problems that demand large size of units.

# 6   acknowledgement

This work is partially supported by AFOSR under the grant 87-0388.

# References

[1] M. Minsky and S. Papert, *Perceptron*, MIT Press Cambridge, Ma(1969).

[2] Y.C. Lee, G. Doolen, H.H. Chen, G.Z. Sun, T. Maxwell, H.Y. Lee and C.L. Giles, *Machine Learning Using A High Order Connection Netweork*, Physica **D22**,776-306 (1986).

[3] H.H. Chen, Y.C. Lee, G.Z. Sun, H.Y. Lee, T. Maxwell and C.L. Giles, *High Order Connection Model For Associate Memory*, AIP Proceedings Vol.151,p.86, Ed. John Denker (1986).

[4] T. Maxwell, C.L. Giles, Y.C. Lee and H.H. Chen, *Nonlinear Dynamics of Artificial Neural System*, AIP Proceedings Vol.151,p.299, Ed. John Denker(1986).

[5] D. Rummenlhart and J. McClelland, *Parallel Distributive Processing*, MIT Press(1986).

[6] L. Breiman, J. Friedman, R. Olshen, C.J. Stone, *Classification and Regression Trees*,Wadsworth Belmont, California(1984).

[7] J.R. Quinlan, *Machine Learning*, Vol.1 No.1(1986).
